# Learning the structure of similarity

**Joshua B. Tenenbaum**
Department of Brain and Cognitive Sciences
Massachusetts Institute of Technology
Cambridge, MA 02139
jbt@psyche.mit.edu

## Abstract

The *additive clustering (ADCLUS)* model (Shepard & Arabie, 1979) treats the similarity of two stimuli as a weighted additive measure of their common features. Inspired by recent work in unsupervised learning with multiple cause models, we propose a new, statistically well-motivated algorithm for discovering the structure of natural stimulus classes using the ADCLUS model, which promises substantial gains in conceptual simplicity, practical efficiency, and solution quality over earlier efforts. We also present preliminary results with artificial data and two classic similarity data sets.

## 1 INTRODUCTION

The capacity to judge one stimulus, object, or concept as *similar* to another is thought to play a pivotal role in many cognitive processes, including generalization, recognition, categorization, and inference. Consequently, modeling subjective similarity judgments in order to discover the underlying structure of stimulus representations in the brain/mind holds a central place in contemporary cognitive science. Mathematical models of similarity can be divided roughly into two families: *spatial* models, in which stimuli correspond to points in a metric (typically Euclidean) space and similarity is treated as a decreasing function of distance; and *set-theoretic* models, in which stimuli are represented as members of salient subsets (presumably corresponding to natural classes or features in the world) and similarity is treated as a weighted sum of common and distinctive subsets.

Spatial models, fit to similarity judgment data with familiar *multidimensional scaling (MDS)* techniques, have yielded concise descriptions of homogeneous, perceptual domains (e.g. three-dimensional color space), often revealing the salient dimensions of stimulus variation (Shepard, 1980). Set-theoretic models are more general, in principle able to accomodate discrete conceptual structures typical of higher-level cognitive domains, as well as dimensional stimulus structures more common in per-

ception (Tversky, 1977). In practice, however, the utility of set-theoretic models is limited by the hierarchical clustering techniques that underlie conventional methods for discovering the discrete features or classes of stimuli. Specifically, hierarchical clustering requires that any two classes of stimuli correspond to disjoint or properly inclusive subsets, while psychologically natural classes may correspond in general to arbitrarily overlapping subsets of stimuli. For example, the subjective similarity of two countries results from the interaction of multiple geographic and cultural factors, and there is no reason a priori to expect the subsets of communist, African, or French-speaking nations to be either disjoint or properly inclusive.

In this paper we consider the *additive clustering (ADCLUS)* model (Shepard & Arabie, 1979), the simplest instantiation of Tversky's (1977) general contrast model that accommodates the arbitrarily overlapping class structures associated with multiple causes of similarity. Here, the similarity of two stimuli is modeled as a weighted additive measure of their common clusters:

$$\hat{s}_{ij} = \sum_{k=1}^{K} w_k f_{ik} f_{jk} + c, \tag{1}$$

where $\hat{s}_{ij}$ is the reconstructed similarity of stimuli $i$ and $j$, the weight $w_k$ captures the salience of cluster $k$, and the binary indicator variable $f_{ik}$ equals 1 if stimulus $i$ belongs to cluster $k$ and 0 otherwise. The additive constant $c$ is necessary because the similarity data are assumed to be on an interval scale.[1] As with conventional clustering models, ADCLUS recovers a system of discrete subsets of stimuli, weighted by salience, and the similarity of two stimuli increases with the number (and weight) of their common subsets. ADCLUS, however, makes none of the structural assumptions (e.g. that any two clusters are disjoint or properly inclusive) which limit the applicability of conventional set-theoretic models. Unfortunately this flexibility also makes the problem of fitting the ADCLUS model to an observed similarity matrix exceedingly difficult.

Previous attempts to fit the model have followed a heuristic strategy to minimize a squared-error energy function,

$$E = \sum_{i \neq j} (s_{ij} - \hat{s}_{ij})^2 = \sum_{i \neq j} (s_{ij} - \sum_k w_k f_{ik} f_{jk})^2, \tag{2}$$

by alternately solving for the best cluster configurations $f_{ik}$ given the current weights $w_k$ and solving for the best weights given the current clusters (Shepard & Arabie, 1979; Arabie & Carroll, 1980). This strategy is appealing because given the cluster configuration, finding the optimal weights becomes a simple linear least-squares problem.[2] However, finding good cluster configurations is a difficult problem in combinatorial optimization, and this step has always been the weak point in previous work. The original ADCLUS (Shepard & Arabie, 1979) and later MAPCLUS (Arabie & Carroll, 1980) algorithms employ ad hoc techniques of combinatorial optimization that sometimes yield unexpected or uninterpretable final results. Certainly, no rigorous theory exists that would explain why these approaches fail to discover the underlying structure of a stimulus set when they do.

Essentially, the ADCLUS model is so challenging to fit because it generates similarities from the interaction of many independent underlying causes. Viewed this way, modeling the structure of similarity looks very similar to the multiple-cause learning

problems that are currently a major focus of study in the neural computation literature (Ghahramani, 1995; Hinton, Dayan, et al., 1995; Saund, 1995; Neal, 1992). Here we propose a novel approach to additive clustering, inspired by the progress and promise of work on multiple-cause learning within the Expectation-Maximization (EM) framework (Ghahramani, 1995; Neal, 1992). Our EM approach still makes use of the basic insight behind earlier approaches, that finding $\{w_k\}$ given $\{f_{ik}\}$ is easy, but obtains better performance from treating the unknown cluster memberships probabilistically as hidden variables (rather than parameters of the model), and perhaps more importantly, provides a rigorous and well-understood theory. Indeed, it is natural to consider $\{f_{ik}\}$ as "unobserved" features of the stimuli, complementing the observed data $\{s_{ij}\}$ in the similarity matrix. Moreover, in some experimental paradigms, one or more of these features may be considered observed data, if subjects report using (or are requested to use) certain criteria in their similarity judgments.

## 2  ALGORITHM

### 2.1  Maximum likelihood formulation

We begin by formulating the additive clustering problem in terms of maximum likelihood estimation with unobserved data. Treating the cluster weights $w = \{w_k\}$ as model parameters and the unobserved cluster memberships $f = \{f_{ik}\}$ as hidden causes for the observed similarities $s = \{s_{ij}\}$, it is natural to consider a hierarchical generative model for the "complete data" (including observed and unobserved components) of the form $p(s, f|w) = p(s|f, w)p(f|w)$. In the spirit of earlier approaches to ADCLUS that seek to minimize a squared-error energy function, we take $p(s|f, w)$ to be gaussian with common variance $\sigma^2$:

$$p(s|f,w) \propto exp\{-\frac{1}{2\sigma^2}\sum_{i \neq j}(s_{ij} - \hat{s}_{ij})^2\} = exp\{-\frac{1}{2\sigma^2}\sum_{i \neq j}(s_{ij} - \sum_k w_k f_{ik} f_{jk})^2\}. \quad (3)$$

Note that $\log p(s|f, w)$ is equivalent to $-E/(2\sigma^2)$ (ignoring an additive constant), where $E$ is the energy defined above. In general, priors $p(f|w)$ over the cluster configurations may be useful to favor larger or smaller clusters, induce a dependence between cluster size and cluster weight, or bias particular kinds of class structures, but only uniform priors are considered here. In this case $-E/(2\sigma^2)$ also gives the "complete data" loglikelihood $\log p(s, f|w)$.

### 2.2  The EM algorithm for additive clustering

Given this probabilistic model, we can now appeal to the EM algorithm as the basis for a new additive clustering technique. EM calls for iterating the following two-step procedure, in order to obtain successive estimates of the parameters $w$ that are guaranteed never to decrease in likelihood (Dempster et al., 1977). In the E-step, we calculate

$$Q(w|w^{(n)}) = \sum_{f'} p(f'|s, w^{(n)}) \log p(s, f'|w) = \frac{1}{2\sigma^2}\langle -E \rangle_{s, w^{(n)}}. \quad (4)$$

$Q(w|w^{(n)})$ is equivalent to the expected value of $E$ as a function of $w$, averaged over all possible configurations $f'$ of the $NK$ binary cluster memberships, given the observed data $s$ and the current parameter estimates $w^{(n)}$. In the M-step, we maximize $Q(w|w^{(n)})$ with respect to $w$ to obtain $w^{(n+1)}$.

Each cluster configuration $f'$ contributes to the mean energy in proportion to its probability under the gaussian generative model in (3). Thus the number of configurations making significant contributions depends on the model variance $\sigma^2$. For large

$\sigma^2$, the probability is spread over many configurations. In the limiting case $\sigma^2 \rightarrow 0$, only the most likely configuration contributes, making EM effectively equivalent to the original approaches presented in Section 1 that use only the single best cluster configuration to solve for the best cluster weights at each iteration.

In line with the basic insight embodied less rigorously in these earlier algorithms, the M-step still reduces to a simple (constrained) linear least-squares problem, because the mean energy $\langle E \rangle = \sum_{i \neq j} \left( s_{ij}^2 - 2s_{ij} \sum_k w_k \langle f_{ik} f_{jk} \rangle + \sum_{kl} w_k w_l \langle f_{ik} f_{jk} f_{il} f_{jl} \rangle \right)$, like the energy $E$, is quadratic in the weights $w_k$. The E-step, which amounts to computing the expectations $m_{ijk} = \langle f_{ik} f_{jk} \rangle$ and $m_{ijkl} = \langle f_{ik} f_{jk} f_{il} f_{jl} \rangle$, is much more involved, because the required sums over all possible cluster configurations $f'$ are intractable for any realistic case. We approximate these calculations using Gibbs sampling, a Monte Carlo method that has been successfully applied to learning similar generative models with hidden variables (Ghahramani, 1995; Neal 1992).[3]

Finally, the algorithm should produce not only estimates of the cluster weights, but also a final cluster configuration that may be interpreted as the psychologically natural features or classes of the relevant domain. Consider the expected cluster memberships $p_{ik} = \langle f_{ik} \rangle_{s,w(n)}$, which give the probability that stimulus $i$ belongs to cluster $k$, given the observed similarity matrix and the current estimates of the weights. Only when all $p_{ik}$ are close to 0 or 1, i.e. when $\sigma^2$ is small enough that all the probability becomes concentrated on the most likely cluster configuration and its neighbors, can we fairly assert which stimuli belong to which classes.

### 2.3   Simulated annealing

Two major computational bottlenecks hamper the efficiency of the algorithm as described so far. First, Gibbs sampling may take a very long time to converge to the equilibrium distribution, particularly when $\sigma^2$ is small relative to the typical energy difference between neighboring cluster configurations. Second, the likelihood surfaces for realistic data sets are typically riddled with local maxima. We solve both problems by annealing on the variance. That is, we run Gibbs sampling using an effective variance $\sigma_{eff}^2$ initially much greater than the assumed model variance $\sigma^2$, and decrease $\sigma_{eff}^2$ towards $\sigma^2$ according to the following two-level scheme. We anneal within the $n^{th}$ iteration of EM to speed the convergence of the Gibbs sampling E-step (Neal, 1993), by lowering $\sigma_{eff}^2$ from some high starting value down to a target $\sigma_{Targ(n)}^2$ for the $n^{th}$ EM iteration. We also anneal between iterations of EM to avoid local maxima (Rose et al., 1990), by intializing $\sigma_{Targ(0)}^2$ at a high value and taking $\sigma_{Targ(n)}^2 \rightarrow \sigma^2$ as $n$ increases.

## 3   RESULTS

In all of the examples below, one run of the algorithm consisted of 100-200 iterations of EM, annealed both within and between iterations. Within each E-step, 10-100 cycles of Gibbs sampling were carried out at the target temperature $\sigma_{Targ}$ while the statistics for $m_{ik}$ and $m_{ijk}$ were recorded. These recorded cycles were preceeded by 20-200 unrecorded cycles, during which the system was annealed from a higher temperature (e.g. $8\sigma_{Targ}^2$) down to $\sigma_{Targ}^2$, to ensure that statistics were collected as close to equilibrium as possible. The precise numbers of recorded and unrecorded iterations were chosen as a compromise between the need for longer samples as the

Table 1: Classes and weights recovered for the integers 0-9.

| Rank | Weight | Stimuli in class | Interpretation |
|------|--------|------------------|----------------|
| 1 | .444 |    2  4     8 | powers of two |
| 2 | .345 | 0 1 2 | small numbers |
| 3 | .331 |     3   6   9 | multiples of three |
| 4 | .291 |          6 7 8 9 | large numbers |
| 5 | .255 |   2 3 4 5 6 | middle numbers |
| 6 | .216 | 1  3  5  7  9 | odd numbers |
| 7 | .214 | 1 2 3 4 | smallish numbers |
| 8 | .172 |     4 5 6 7 8 | largish numbers |

Variance accounted for = 90.9% with 8 clusters (additive constant = .148).

number of hidden variables is increased and the need to keep computation times practical.

## 3.1 Artificial data

We first report results with artificial data, for which the true cluster memberships and weights are known, to verify that the algorithm does in fact find the desired structure. We generated 10 data sets by randomly assigning each of 12 stimuli independently and with probability 1/2 to each of 8 classes, and choosing random weights for the classes uniformly from $[0.1, 0.6]$. These numbers are grossly typical of the real data sets we examine later in this section. We then calculated the observed similarities from (1), added a small amount of random noise (with standard deviation equal to 5% of the mean noise-free similarity), and symmeterized the similarity matrix.

The crucial free parameter is $K$, the assumed number of stimulus classes. When the algorithm was configured with the correct number of clusters ($K = 8$), the original classes and weights were recovered during the first run of the algorithm on all 10 data sets, after an average of 58 EM iterations (low 30, high 92). When the algorithm was configured with $K = 7$ clusters, one less than the correct number, the seven classes with highest weight were recovered on 9/10 first runs. On these runs, the recovered weights and true weights had a mean correlation of 0.948 ($p < .05$ on each run). When configured with $K = 5$, the first run recovered either four of the top five classes (6/10 trials) or three of the top five (4/10 trials). When configured with too many clusters ($K = 12$), the algorithm typically recovered only 8 clusters with significantly non-zero weights, corresponding to the 8 correct classes. Comparable results are not available for ADCLUS or MAPCLUS, but at least we can be satisfied that our algorithm achieves a basic level of competence and robustness.

## 3.2 Judged similarities of the integers 0-9

Shepard et al. (1975) had subjects judge the similarities of the integers 0 through 9, in terms of the "abstract concepts" of the numbers. We analyzed the similarity matrix (Shepard, personal communication) obtained by pooling data across subjects and across three conditions of stimulus presentation (verbal, written-numeral, and written-dots). We chose this data set because it illustrates the power of additive clustering to capture a complex, overlapping system of classes, and also because it serves to compare the performance of our algorithm with the original ADCLUS algorithm. Observe first that two kinds of classes emerge in the solution. Classes 1, 3, and 6 represent familiar arithmetic concepts (e.g. "multiples of three", "odd numbers"), while the remaining classes correspond to subsets of consecutive integers

Table 2: Classes and weights recovered for the 16 consonant phonemes.

| Rank | Weight | Stimuli in class | Interpretation |
|------|--------|------------------|----------------|
| 1 | .800 | f θ | front unvoiced fricatives |
| 2 | .572 | d g | back voiced stops |
| 3 | .463 | p k | unvoiced stops (omitting t) |
| 4 | .424 | b v ð | front voiced |
| 5 | .357 | p t k | unvoiced stops |
| 6 | .292 | m n | nasals |
| 7 | .169 | d g v ð z ẑ | voiced (omitting b) |
| 8 | .132 | p t k f θ s | unvoiced (omitting š) |

Variance accounted for = 90.2% with 8 clusters (additive constant = .047).

and thus together represent the dimension of numerical magnitude. In general, both arithmetic properties and numerical magnitude contribute to judged similarity, as every number has features of both types (e.g. 9 is a "large" "odd" "multiple of three"), except for 0, whose only property is "small." Clearly an overlapping clustering model is necessary here to accomodate the multiple causes of similarity.

The best solution reported for these data using the original ADCLUS algorithm consisted of 10 classes, accounting for 83.1% of the variance of the data (Shepard & Arabie, 1979).[4] Several of the clusters in this solution differed by only one or two members (e.g. three of the clusters were {0,1}, {0,1,2}, and {0,1,2,3,4}), which led us to suspect that a better fit might be obtained with fewer than 10 classes. Table 2 shows the best solution found in five runs of our algorithm, accounting for 90.9% of the variance with eight classes. Compared with our solution, the original ADCLUS solution leaves almost twice as much residual variance unaccounted for, and with 10 classes, is also less parsimonious.

## 3.3   Confusions between 16 consonant phonemes

Finally, we examine Miller & Nicely's (1955) classic data on the confusability of 16 consonant phonemes, collected under varying signal/noise conditions with the original intent of identifying the features of English phonology (compiled and reprinted in Carroll & Wish, 1974). Note that the recovered classes have reasonably natural interpretations in terms of the basic features of phonological theory, and a very different overall structure from the classes recovered in the previous example. Quite significantly, the classes respect a hierarchical structure almost perfectly, with class 3 included in class 5, classes 1 and 5 included in class 8, and so on. Only the absence of /b/ in class 7 violates the strict hierarchy.

These data also provide the only convenient oppportunity to compare our algorithm with the MAPCLUS approach to additive clustering (Arabie & Carroll, 1980). The published MAPCLUS solution accounts for 88.3% of the variance in this data, using eight clusters. Arabie & Carroll (1980) report being "substantively perturbed" (p. 232) that their algorithm does not recover a distinct cluster for the nasals /m n/, which have been considered a very salient subset in both traditional phonology (Miller & Nicely, 1955) and other clustering models (Shepard, 1980). Table 3 presents our eight-cluster solution, accounting for 90.2% of the variance. While this represents only a marginal improvement, our solution does contain a cluster for the nasals, as expected on theoretical grounds.

## 3.4 Conclusion

These examples show that ADCLUS can discover meaningful representations of stimuli with arbitrarily overlapping class structures (arithmetic properties), as well as dimensional structure (numerical magnitude) or hierarchical structure (phoneme families) when appropriate. We have argued that modeling similarity should be a natural application of learning generative models with multiple hidden causes, and in that spirit, presented a new probabilistic formulation of the ADCLUS model and an algorithm based on EM that promises better results than previous approaches. We are currently pursuing several extensions: enriching the generative model, e.g. by incorporating significant prior structure, and improving the fitting process, e.g. by developing efficient and accurate mean field approximations. More generally, we hope this work illustrates how sophisticated techniques of computational learning can be brought to bear on foundational problems of structure discovery in cognitive science.

## Acknowledgements

I thank P. Dayan, W. Richards, S. Gilbert, Y. Weiss, A. Hershowitz, and M. Bernstein for many helpful discussions, and Roger Shepard for generously supplying inspiration and unpublished data. The author is a Howard Hughes Medical Institute Predoctoral Fellow.

## Footnotes

[1]In the remainder of this paper, we absorb $c$ into the sum over $k$, taking the sum over $k = 0, \ldots, K$, defining $w_0 \equiv c$, and fixing $f_{i0} = 1, (\forall i)$.

[2]Strictly speaking, because the weights are typically constrained to be nonnegative, more elaborate techniques than standard linear least-squares procedures may be required.

[3]We generally also approximate $m_{ijkl} \approx m_{ijk} m_{ijl}$, which usually yields satisfactory results with much greater efficiency.

[4]Variance accounted for = $1 - E/\sum_{i \neq j}(s_{ij} - \bar{s})^2$, where $\bar{s}$ is the mean of the set $\{s_{ij}\}$.

## References

Arabie, P. & Carroll, J. D. (1980). MAPCLUS: A mathematical programming approach to fitting the ADCLUS model. *Psychometrika* **45**, 211-235.

Carroll, J. D. & Wish, M. (1974) Multidimensional perceptual models and measurement methods. In *Handbook of Perception, Vol. 2*. New York: Academic Press, 391-447.

Dempster, A. P., Laird, N. M., & Rubin, D. B. (1977). Maximum likelihood estimation from incomplete data via the EM Algorithm (with discussion). *J. Roy. Stat. Soc.* **B39**, 1-38.

Ghahramani, Z. (1995). Factorial learning and the EM algorithm. In G. Tesauro, D. S. Touretzky, & T. K. Leen (eds.), *Advances in Neural Information Processing Systems 7*. Cambridge, MA: MIT Press, 617-624.

Hinton, G. E., Dayan, P., Frey, B. J., & Neal, R. M. (1995) The "wake-sleep" algorithm for unsupervised neural networks. *Science* **268**, 1158-1161.

Miller, G. A. & Nicely, P. E. (1955). An analysis of perceptual confusions among some English consonants. *J. Ac. Soc. Am.* **27**, 338-352.

Neal, R. M. (1992). Connectionist learning of belief networks. *Artif. Intell.* **56**, 71-113.

Neal, R. M. (1993). Probabilistic inference using Markov chain Monte Carlo methods. Technical Report CRG-TR-93-1, Dept. of Computer Science, U. of Toronto.

Rose, K., Gurewitz, F., & Fox, G. (1990). Statistical mechanics and phase transitions in clustering. *Physical Review Letters* **65**, 945-948.

Saund, E. (1995). A multiple cause mixture model for unsupervised learning. *Neural Computation* **7**, 51-71.

Shepard, R. N. & Arabie, P. (1979). Additive clustering: Representation of similarities as combinations of discrete overlapping properties. *Psychological Review* **86**, 87-123.

Shepard, R. N., Kilpatric, D. W., & Cunningham, J. P., (1975). The internal representation of numbers. *Cognitive Psychology* **7**, 82-138.

Shepard, R. N. (1980). Multidimensional scaling, tree-fitting, and clustering. *Science* **210**, 390-398.

Tversky, A. (1977). Features of similarity. *Psychological Review* **84**, 327-352.